# Minimization of Continuous Bethe Approximations: A Positive Variation

**Jason L. Pacheco** and **Erik B. Sudderth**
Department of Computer Science, Brown University, Providence, RI
{pachecoj,sudderth}@cs.brown.edu

## Abstract

We develop convergent minimization algorithms for Bethe variational approximations which explicitly constrain marginal estimates to families of valid distributions. While existing message passing algorithms define fixed point iterations corresponding to stationary points of the Bethe free energy, their greedy dynamics do not distinguish between local minima and maxima, and can fail to converge. For continuous estimation problems, this instability is linked to the creation of invalid marginal estimates, such as Gaussians with negative variance. Conversely, our approach leverages multiplier methods with well-understood convergence properties, and uses bound projection methods to ensure that marginal approximations are valid at all iterations. We derive general algorithms for discrete and Gaussian pairwise Markov random fields, showing improvements over standard loopy belief propagation. We also apply our method to a hybrid model with both discrete and continuous variables, showing improvements over expectation propagation.

## 1 Introduction

Variational inference algorithms pose probabilistic inference as an optimization over distributions. Typically the optimization is formulated by minimizing an objective known as the *Gibbs free energy* [1]. Variational methods relax an otherwise intractable optimal inference problem by approximating the entropy-based objective, and considering appropriately simplified families of approximating distributions [2]. Local message passing algorithms offer a computationally efficient method for extremizing variational free energies. Loopy belief propagation (LBP), for example, optimizes a relaxed objective known as the *Bethe free energy* [1, 2], which we review in Sec. 2. Expectation propagation (EP) [3] is a generalization of LBP which shares the same objective, but optimizes over a relaxed set of constraints [4] applicable to a broader family of continuous inference problems.

In general, neither LBP nor EP are guaranteed to converge. Even in simple continuous models, both methods may improperly estimate invalid or degenerate marginal distributions, such as Gaussians with negative variance. Such degeneracy typically occurs in classes of models for which convergence properties are poor, and there is evidence that these problems are related [5, 6],

Extensive work has gone into developing algorithms which improve on LBP for models with discrete variables, for example by bounding [7, 8] or convexifying [9] the free energy objective. Gradient optimization methods have been applied successfully to binary Ising models [10], but when applied to Gaussian models this approach suffers similar non-convergence and degeneracy issues as LBP. Work on optimization of continuous variational free energies has primarily focused on addressing convergence problems [11]. None of these approaches directly address degeneracy in the continuous case, and computation may be prohibitively expensive for these direct minimization schemes.

By leveraging gradient projection methods from the extensive literature on constrained nonlinear optimization, we develop an algorithm which ensures that marginal estimates remain valid and normalizable at all iterations. In doing so, we account for important constraints which have been ignored

by previous variational derivations of the expectation propagation algorithm [12, 6, 11]. Moreover, by adapting the method of multipliers [13], we guarantee that our inference algorithm converges for most models of practical interest.

We begin by introducing the Bethe variational problem (Sec. 2). We briefly review the correspondence between the Lagrangian formalism and message passing and discuss implicit normalizability assumptions which, when violated, lead to degeneracy in message passing algorithms. We discuss the method of multipliers, gradient projection, and convergence properties (Sec. 3). We then provide derivations (Sec. 4) for discrete MRFs, Gaussian MRFs, and hybrid models with potentials defined by discrete mixtures of Gaussian distributions. Experimental results in Sec. 5 demonstrate substantial improvements over baseline message passing algorithms.

## 2 Bethe Variational Problems

For simplicity, we restrict our attention to pairwise Markov random fields (MRF) [2], with graphs $G(\mathcal{V}, \mathcal{E})$ defined by nodes $\mathcal{V}$ and undirected edges $\mathcal{E}$. The joint distribution then factorizes as

$$p(x) = \frac{1}{Z_p} \prod_{s \in \mathcal{V}} \varphi_s(x_s) \prod_{(s,t) \in \mathcal{E}} \varphi_{st}(x_s, x_t) \tag{1}$$

for some non-negative *potential functions* $\varphi(\cdot)$. Often this distribution is a posterior given fixed observations $y$, but we suppress this dependence for notational simplicity. We are interested in computing the *log partition function* $\log Z_p$, and/or the marginal distributions $p(x_s), s \in \mathcal{V}$.

Let $q(x; \mu)$ denote an *exponential family* of densities with sufficient statistics $\phi(x) \in \mathbb{R}^d$:

$$q(x; \mu) \propto \exp\{\theta^T \phi(x)\}, \; \mu = \mathbb{E}_q[\phi(x)]. \tag{2}$$

To simplify subsequent algorithm development, we index distributions via their *mean parameters* $\mu$. We associate each node $s \in \mathcal{V}$ with an exponential family $q_s(x_s; \mu_s)$, $\phi_s(x) \in \mathbb{R}^{d_s}$, and each edge $(s,t) \in \mathcal{E}$ with a family $q_{st}(x_s, x_t; \mu_{st})$, $\phi_{st}(x) \in \mathbb{R}^{d_{st}}$. Because $q_s(x_s; \mu_s)$ is a valid probability distribution, $\mu_s$ must lie in a set of *realizable* mean parameters, $\mu_s \in \mathcal{M}_s$. Similarly, $\mu_{st} \in \mathcal{M}_{st}$. For example, $\mathcal{M}_s$ and $\mathcal{M}_{st}$ might require Gaussians to have positive semidefinite covariances.

We can express the log partition as the solution to an optimization problem,

$$-\log Z_p = \min_{\mu \in \mathbb{M}(G)} \mathbb{E}_\mu[-\log p(x)] - \mathcal{H}[\mu] = \min_{\mu \in \mathbb{M}(G)} \mathcal{F}(\mu), \tag{3}$$

where $\mathcal{H}[\mu]$ is the entropy of $q(x; \mu)$, $\mathbb{E}_\mu[\cdot]$ denotes expectation with respect to $q(x; \mu)$, and $\mathcal{F}(\mu)$ is known as the *variational free energy*. Mean parameters $\mu$ lie in the *marginal polytope* $\mathbb{M}(G)$ if and only if there exists some valid, joint probability distribution with those moments.

Exactly characterizing $\mathbb{M}(G)$ may require exponentially many constraints, so we relax the optimization to be over a set of *locally consistent* marginal distributions $\mathbb{L}(G)$, which are properly normalized and satisfy expectation constraints associated with each edge of the graph:

$$C_s(\mu) = 1 - \int q_s(x_s; \mu_s) \, dx_s, \quad C_{ts}(\mu) = \mu_s - \mathbb{E}_{q_{st}}[\phi_s(x_s)]. \tag{4}$$

This is a relaxation in the sense that $\mathbb{M}(G) \subset \mathbb{L}(G)$ with strict equality if $G$ does not contain cycles. We approximate the entropy $\mathcal{H}[\mu]$ with the entropy of a tree-structured distribution $q(x; \mu)$. Such an approximation is tractable and consistent with $\mathbb{L}(G)$, and yields the *Bethe free energy*:

$$\mathcal{F}_B(\mu) = \sum_{(s,t) \in \mathcal{E}} \mathbb{E}_{q_{st}}[\log q_{st}(x_s, x_t; \mu_{st}) - \psi_{st}(x_s, x_t)] - \sum_{s \in \mathcal{V}} (n_s - 1) \mathbb{E}_{q_s}[\log q_s(x_s; \mu_s) - \varphi_s(x_s)] \tag{5}$$

Here, $\psi_{st}(\cdot) = \varphi_{st}(\cdot)\varphi_s(\cdot)\varphi_t(\cdot)$, and the mean parameters $\mu$ are valid within the constraint set $\mathcal{M} = \bigcup_s \mathcal{M}_s \bigcup_{st} \mathcal{M}_{st}$. The resulting objective is the *Bethe variational problem* (BVP):

$$\begin{aligned} \underset{\mu}{\text{minimize}} \quad & \mathcal{F}_B(\mu) \\ \text{subject to} \quad & C_{ts}(\mu) = 0, \forall s \in \mathcal{V}, t \in N(s) \\ & C_s(\mu) = 0, \forall s \in \mathcal{V}, \\ & \{\mu_s : s \in \mathcal{V}\} \cup \{\mu_{st} : (s,t) \in \mathcal{E}\} \in \mathcal{M}. \end{aligned} \tag{6}$$

Here, $N(s)$ denotes the set of neighbors of node $s \in \mathcal{V}$.

## 2.1 Correspondence to Message Passing

We can optimize the BVP (6) by relaxing the normalization and local consistency constraints with Lagrange multipliers. Constrained minima are characterized by stationary points of the Lagrangian,

$$\mathcal{L}(x, \lambda) = \mathcal{F}_B(q) + \sum_s \lambda_s C_s + \sum_s \sum_{t \in N(s)} \lambda_{ts} C_{ts}. \tag{7}$$

Equivalence between LBP fixed points and stationary points of the Lagrangian for the discrete case have been discussed extensively [1, 2]. Similar correspondence has been shown more generally for EP fixed points [2, 4]. Since our focus is on the continuous case we briefly review the correspondence between Gaussian LBP fixed points and the Gaussian Bethe free energy. For simplicity we focus on zero-mean $p(x) = N(x \mid 0, J^{-1})$, where diagonal precision entries $J_{ss} = A_s$ and

$$\varphi_s(x_s) = \exp\left\{-\frac{1}{2}x_s^2 A_s\right\}, \qquad \varphi_{st}(x_s, x_t) = \exp\left\{-\frac{1}{2}\begin{pmatrix} x_s & x_t \end{pmatrix} \begin{pmatrix} 0 & J_{st} \\ J_{st} & 0 \end{pmatrix} \begin{pmatrix} x_s \\ x_t \end{pmatrix}\right\}.$$

Let $q(x_s) = N(x_s \mid 0, V_s)$, $q(x_s, x_t) = N(\begin{pmatrix} x_s \\ x_t \end{pmatrix} \mid 0, \Sigma_{st})$, $\Sigma_{st} = \begin{pmatrix} V_{ts} & P_{st} \\ P_{ts} & V_{st} \end{pmatrix}$, and $\widetilde{B}_{st} = \begin{pmatrix} A_s & J_{st} \\ J_{st} & A_t \end{pmatrix}$. The Gaussian Bethe free energy then equals

$$\mathcal{F}_{GB}(V, \Sigma) = \frac{1}{2} \sum_{(s,t) \in \mathcal{E}} \left( \text{tr}(\Sigma_{st} \widetilde{B}_{st}) - \log|\Sigma_{st}| \right) - \sum_{s \in \mathcal{V}} \left( \frac{n_s - 1}{2} \right) (V_s A_s - \log V_s). \tag{8}$$

The locally consistent marginal polytope $\mathbb{L}(G)$ consists of the constraints $C_{ts}(V) = V_s - V_{ts}$ for all nodes $s \in \mathcal{V}$ and edges $(s, t) \in \mathcal{E}$. The Lagrangian is given by,

$$\mathcal{L}(V, \Sigma, \lambda) = \mathcal{F}_{GB}(V, \Sigma) + \sum_s \sum_{t \in N(s)} \lambda_{ts} [V_s - V_{ts}]. \tag{9}$$

Taking the derivative with respect to the node marginal variance $\frac{\partial \mathcal{L}}{\partial V_s} = 0$ yields the stationary point $V_s^{-1} = A_s + \frac{1}{n_s - 1} \sum_{t \in N(s)} \lambda_{ts}$. For a Gaussian LBP algorithm with messages parametrized as $m_{t \to s}(x_s) = \exp\left\{-\frac{1}{2}x_s^2 \Lambda_{t \to s}\right\}$, fixed points of the node marginal precision are given by

$$\Lambda_s = A_s + \sum_{t \in N(s)} \Lambda_{t \to s}$$

Let $\lambda_{ts} = -\frac{1}{2} \sum_{a \in N(s) \backslash t} \Lambda_{a \to s}$. Substituting back into the stationary point conditions yields $V_s^{-1} \Rightarrow \Lambda_s$. A similar construction holds for the pairwise marginals. Inverting the correspondence between multipliers and message parameters yields the converse $V_s^{-1} \Leftarrow \Lambda_s$ (c.f. [4]).

## 2.2 Message Passing Non-Convergence and Degeneracy

While local message passing algorithms are convenient for many applications, their convergence is not guaranteed in general. In particular, LBP often fails to converge for networks with tight loops [1] such as the $3 \times 3$ lattice of Figure 1(a). For non-Gaussian models with continuous variables, convergence of the EP algorithm can be even more problematic [11].

For continuous models message updates may yield degenerate, unnormalizable marginal distributions which do not correspond to stationary points of the Lagrangian. For example, for Gaussian MRFs the Bethe free energy $\mathcal{F}_B(\cdot)$ in (5) is derived from expectations with respect to variational distributions $q_s(x_s; \mu_s), q_{st}(x_s, x_t; \mu_{st})$. If a set of hypothesized marginals are not normalizable (positive variance), the Gaussian Bethe free energy $\mathcal{F}_{GB}(\cdot)$ is invalid and undefined.

Degenerate marginals arise because the constraint set $\mathcal{M}$ is not represented in the Lagrangian (7); this issue is mentioned briefly in [2] but is not dealt with computationally. Figure 1(b) demonstrates this issue for a simple, three-node Gaussian MRF. Here LBP produces marginal variances which oscillate between impossibly large positive, and non-sensical negative, values. Such degeneracies are arguably more problematic for EP since its *moment matching* steps require expected values with respect to an *augmented distribution* [3], which may involve an unbounded integral.

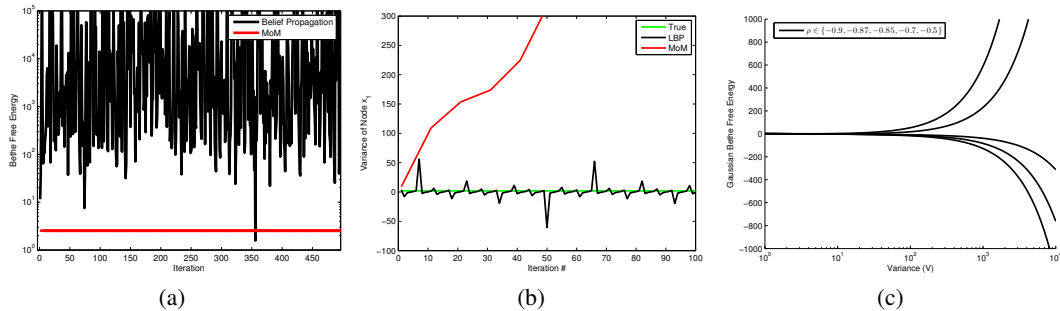

Figure 1: (a) Bethe free energy versus iteration for 3x3 toroidal binary MRF. (b) Node marginal variance estimates per iteration for a symmetric, single-cycle Gaussian MRF with three nodes (plot is of $x_1$, other nodes are similar). (c) For the model from (b), the Gaussian Bethe free energy is unbounded on the constraint set.

## 2.3 Unboundedness of the Gaussian Bethe Free Energy

Conditions under which the simple LBP and EP updates are guaranteed to be accurate are of great practical interest. For Gaussian MRFs, the class of *pairwise normalizable* models are sufficient to guarantee LBP stability and convergence [5]. For non-pairwise normalizable models the Gaussian Bethe free energy is unbounded below [6] on the set of local consistency constraints $\mathbb{L}(G)$.

We offer a small example consisting of a non-pairwise normalizable symmetric single cycle with 3 nodes. Diagonal precision elements are $J_{ss} = 1.0$, and off-diagonal elements $J_{st} = 0.6$. We embed marginalization constraints into a symmetric parametrization $V_s = V$ and $\Sigma_{st} = \begin{pmatrix} V & \rho V \\ \rho V & V \end{pmatrix}$. Feasible solutions within the constraint set are characterized by $V > 0$ and $-1 < \rho < 1$. Substituting this parametrization into the Gaussian free energy (8), and performing some simple algebra, yields

$$\mathcal{F}_{GB}(V, \rho) = -\frac{3}{2}\log V + \frac{3}{2}V(1 + 1.2\rho) - \frac{3}{2}\log(1 - \rho^2).$$

For $\rho < -\frac{1}{1.2}$ the free energy is unbounded below at rate $\mathcal{O}(-V)$. Figure 1(c) illustrates the Bethe free energy for this model as a function of $V$, and for several values of $\rho$.

More generally, it has been shown that Gaussian EP messages are always normalizable (positive variance) for models with log-concave potentials [14]. It has been conjectured, but not proven, that EP is also guaranteed to converge for such models [15]. For Gaussian MRFs, we note that the family of log-concave models coincides with the pairwise normalizability condition. Our work seeks to improve inference for non-log-concave models with bounded Bethe free energies.

## 3 Method of Multipliers

Given our complete constrained formulation of the Bethe variational problem, we avoid convergence and degeneracy problems via direct minimization using the *method of multipliers* (MoM) [13]. In general terms, given some convex feasible region $\mathcal{M}$, consider the equality constrained problem

$$\underset{x \in \mathcal{M}}{\text{minimize}} \ f(x) \quad \text{subject to} \quad h(x) = 0$$

With penalty parameter $c > 0$, we form the *augmented Lagrangian* function,

$$\mathcal{L}_c(x, \lambda) = f(x) + \lambda^T h(x) + \frac{1}{2}c||h(x)||^2 \qquad (10)$$

Given a multiplier vector $\lambda_k$ and penalty parameter $c_k$ we update the primal and dual variables as,

$$x_k = \arg\min_{x \in \mathcal{M}} \mathcal{L}_{c_k}(x, \lambda_k), \quad \lambda_{k+1} = \lambda_k + c_k h(x_k).$$

The penalty multiplier can be updated as $c_{k+1} \geq c_k$ according to some fixed update schedule, or based on the results of the optimization step. An update rule that we find useful [13] is to increase the penalty parameter by $\beta > 1$ if the constraint violation is not improved by a factor $0 < \gamma < 1$ over the previous iteration,

$$c_{k+1} = \begin{cases} \beta c_k & \text{if } \|h(x_k)\| > \gamma \|h(x_{k-1})\|, \\ c_k & \text{if } \|h(x_k)\| \leq \gamma \|h(x_{k-1})\|. \end{cases}$$

## 3.1 Gradient Projection Methods

The augmented Lagrangian $\mathcal{L}_c(x, \lambda)$ is a partial one, where feasibility of mean parameters ($x \in \mathcal{M}$) is enforced explicitly by projection. A simple *gradient projection* method [13] defines a sequence

$$x_{k+1} = x_k + \alpha_k(\bar{x}_k - x_k), \qquad \bar{x}_k = [x_k - s_k \nabla f(x_k)]^+ .$$

The notation $[\cdot]^+$ denotes a projection onto the constraint set $\mathcal{M}$. After taking a step $s_k > 0$ in the direction of the negative gradient, we project the result onto the constraint set to obtain a feasible direction $\bar{x}_k$. We then compute $x_{k+1}$ by taking a step $\alpha_k \in (0, 1]$ in the direction of $(\bar{x}_k - x_k)$. If $x_k - s_k \nabla f(x_k)$ is feasible, gradient projection reduces to unconstrained steepest descent.

There are multiple such projection steps in each *inner-loop* iteration of MoM (e.g. each $x_k$ update). For our experiments we use a projected quasi-Newton method [16] and step-sizes $\alpha_k$ and $s_k$ are chosen using an Armijo rule [13, Prop. 2.3.1].

## 3.2 Convergence of Multiplier Methods

Convergence and rate of convergence results have been proven [17, Proposition 2.4] for the Method of Multipliers with a quadratic penalty and multiplier iteration $\lambda_{k+1} = \lambda_k + c_k h(x_k)$. The main regularity assumptions are that the sequence $\{\lambda_k\}$ is bounded, and there is a local minimum for which a Lagrange multiplier pair $(x^*, \lambda^*)$ exists satisfying second-order sufficiency conditions, so that $\nabla_x \mathcal{L}_0(x^*, \lambda^*) = 0$ and $z^T \nabla_{xx}^2 \mathcal{L}_0(x^*, \lambda^*) z > 0$ for all $z \neq 0$. It then follows that there exists some $\bar{c}$ such that for all $c \geq \bar{c}$, the augmented Lagrangian also contains a strict local minimum $z^T \nabla_{xx}^2 \mathcal{L}_c(x^*, \lambda^*) z > 0$.

For convergence, the initialization of the Lagrange multiplier $\lambda_0$ and penalty parameter $c_0$ must be such that $\|\lambda_0 - \lambda^*\| < \delta c_0$ for some $\delta > 0$ and $c \geq \bar{c}$ which depend on the objective and constraints. In practice, a poor initialization of the multiplier $\lambda_0$ can often be offset by a sufficiently high $c_0$. A final technical note is that convergence proofs assume the sequence of unconstrained optimizations which yield $x_k$ stays in the neighborhood of $x^*$ after some $k$. This does not hold in general, but can be encouraged by warm-starting the unconstrained optimization with the previous $x_{k-1}$.

To invoke existing convergence results we must show that a local minimum $x^*$ exists for each of the free energies we consider; a sufficient condition is then that the Bethe free energy is bounded from below. This property has been previously established for general discrete MRFs [18], for pairwise normalizable Gaussian MRFs [6], and for the clutter model [3]. For non-pairwise normalizable Gaussian MRFs, the example of Section 2.3 shows that the Bethe variational objective is unbounded below, and further may not contain any local optima. While the method of multipliers does not converge in this situation, its non-convergence is due to fundamental flaws in the Bethe approximation.

## 4 MoM Algorithms for Probabilistic Inference

We derive MoM algorithms which minimize the Bethe free energy for three different families of graphical models. For each model we define the form of the joint distribution, Bethe free energy (5), local consistency constraints, augmented Lagrangian, and the gradient projection step. Gradients, which can be notationally cumbersome, are given in the supplemental material.

### 4.1 Gaussian Markov Random Fields

We have already introduced the Lagrangian (9) for the Gaussian MRF. The Gaussian Bethe free energy (8) is always unbounded below off of the constraint set in node marginal variances $V_s$. We correct this by adding an additional fixed penalty in the augmented Lagrangian,

$$\mathcal{L}_c(V, \Sigma, \lambda) = \mathcal{F}_{GB}(V) + \sum_s \sum_{t \in N(s)} \lambda_{ts} [V_s - V_{ts}]$$

$$+ \frac{\kappa}{2} \sum_s \sum_{t \in N(s)} [\log V_s - \log V_{ts}]^2 + \frac{c}{2} \sum_s \sum_{t \in N(s)} [V_s - V_{ts}]^2 .$$

We keep $\kappa \geq 1$ fixed so that existing convergence theory remains applicable. The set of realizeable mean parameters $\mathcal{M}$ is the set of symmetric positive semidefinite matrices $V_s, \Sigma_{st}$. We therefore

must solve a series of constrained optimizations of the form, $\min_{V,\Sigma} \mathcal{L}_{c_k}(V, \Sigma, \lambda_k)$, subject to $V_s \geq 0, \Sigma_{st} \succeq 0$. The gradient projection step is easily expressed in terms of correlation coefficients $\rho_{st}$,

$$\Sigma_{st} = \begin{bmatrix} V_{st} & \rho_{st}\sqrt{V_{st}V_{ts}} \\ \rho_{st}\sqrt{V_{st}V_{ts}} & V_{ts} \end{bmatrix}.$$

Then, $\Sigma_{st} \succeq 0$ if and only if $V_{st} \geq 0$, $V_{ts} \geq 0$, and $-1 \leq \rho_{st} \leq 1$. The projection step is then,

$$V_{st} = \max(0, V_{st}), \quad V_{ts} = \max(0, V_{ts}), \quad \rho_{st} = \max(-1, \min(1, \rho_{st})).$$

The full MoM algorithm then follows from gradient derivations in the supplemental material.

Recall that in Section 2.3, we showed that the Gaussian Bethe free energy is unbounded on the constraint set for non-pairwise normalizable models. We run MoM on the symmetric three-node cycle from this discussion and find that MoM, correctly, identifies an unbounded direction, and Figure 1(b) shows that the node marginal variances indeed diverge to infinity.

## 4.2 Discrete Markov Random Fields

Consider a discrete MRF where all variables $x_s \in \mathcal{X}_s = \{1, \ldots, K_s\}$. The variational marginal distributions are then $q_s(x_s; \tau) = \prod_{k=1}^{K_s} \tau(x_s)^{\mathbb{I}(x_s, k)}$, and have mean parameters $\tau \in \mathbb{R}^{K_s}$. Let $\tau(x_s)$ denote element $x_s$ of vector $\tau$. Pairwise marginals have mean parameters $\tau_{st} \in \mathbb{R}^{K_s \times K_t}$ similarly indexed as $\tau_{st}(x_s, x_t)$. The discrete Bethe free energy is then

$$\mathcal{F}_B(\tau; \varphi) = \sum_{(s,t)\in\mathcal{E}} \sum_{x_s} \sum_{x_t} \tau_{st}(x_s, x_t)[\log \tau_{st}(x_s, x_t) - \log \phi_{st}(x_s, x_t)]$$
$$- \sum_{s\in\mathcal{V}} \sum_{x_s} (n_s - 1)\tau_s(x_s)[\log \tau_s(x_s) - \log \varphi_s(x_s)].$$

For this discrete model, our expectation constraints reduce to the following normalization and marginalization constraints:

$$C_s(\tau) = 1 - \sum_{x_s} \tau_s(x_s), \quad C_{ts}(x_s; \tau) = \tau_s(x_s) - \sum_{x_t} \tau_{st}(x_s, x_t).$$

The augmented Lagrangian is then,

$$\mathcal{L}_c(\tau, \lambda, \xi; \varphi) = \mathcal{F}_B(\tau; \varphi) + \sum_{(s,t)\in\mathcal{E}} \left[\sum_{x_s} \lambda_{ts}(x_s)C_{ts}(x_s; \tau) + \sum_{x_t} \lambda_{st}(x_t)C_{st}(x_t; \tau)\right] \quad (11)$$
$$+ \sum_{s\in\mathcal{V}} \xi_{ss}C_s(\tau) + \frac{c}{2}\sum_{s\in\mathcal{V}} C_s(\tau)^2 + \frac{c}{2}\sum_{(s,t)\in\mathcal{E}} \left[\sum_{x_s} C_{ts}(x_s; \tau)^2 + \sum_{x_t} C_{st}(x_t; \tau)^2\right].$$

Mean parameters must be non-negative to be valid, so $\mathcal{M} = \{\tau_s, \tau_{st} : \tau_s \geq 0, \tau_{st} \geq 0\}$. This constraint is enforced by a bound projection $\tau_s(x_s) = \max(0, \tau_s(x_s))$, and similarly for the pairwise marginals. While these constraints are never active in BP fixed point iterations, they must be enforced in gradient optimization. With these pieces and the gradient computations presented in the supplement, implementation of MoM optimization for the discrete MRF is straightforward.

## 4.3 Discrete Mixtures of Gaussian Potentials

We are particularly interested in tractable inference in hybrid models with discrete and conditionally Gaussian random variables. A simple example of such a model is the *clutter problem* [3], whose joint distribution models $N$ conditionally independent Gaussian observations $\{y_i\}_{i=1}^N$. These observations may either be centered on a target scalar $x \in \mathbb{R}$ ($z_i = 1$) or drawn from a background clutter distribution ($z_i = 0$). If target observations occur with frequency $\beta_0$, we then have

$$x \sim N(\mu_0, P_0), \quad z_i \sim \text{Ber}(\beta_0), \quad y_i \mid x, z_i \sim N(0, \sigma_0^2)^{(1-z_i)} N(x, \sigma_1^2)^{z_i}$$

The corresponding variational posterior distributions are,

$$q_0(x) = N(m_0, V_0), \quad q_i(x, z_i) = ((1-\beta_i)N(x \mid m_{i0}, V_{i0}))^{(1-z_i)} (\beta_i N(x \mid m_{i1}, V_{i1}))^{z_i}.$$

Assuming normalizable marginals with $V_0 \geq 0$, $V_{i0} \geq 0$, $V_{i1} \geq 0$, as always ensured by our multiplier method, we define the Bethe free energy $\mathcal{F}_{CGB}(m, V, \beta)$ in terms of the mean parameters in the supplemental material. Expectation constraints are given by,

$$C_i^{\text{mean}} = \mathbb{E}_0[x] - \mathbb{E}_i[x], \quad C_i^{\text{var}} = \text{Var}_0[x] - \text{Var}_i[x],$$

where $\mathbb{E}_i[\cdot]$ and $\text{Var}_i[\cdot]$ denote the mean and variance of the Gaussian mixture $q_i(x, z_i)$. Combining the free energy, constraints, and additional positive semidefinite constraints on the marginal variances we have the BVP for the clutter model,

$$
\begin{aligned}
\underset{m, V, \beta}{\text{minimize}} \quad & \mathcal{F}_{CGB}(m, V, \beta; \varphi) \\
\text{subject to} \quad & C_i^{\text{mean}} = 0, \ C_i^{\text{var}} = 0, \ \text{for all } i = 1, 2, \ldots, N \\
& V_0 \geq 0, V_{i0} \geq 0, V_{i1} \geq 0
\end{aligned}
\tag{12}
$$

Derivation of the free energy and augmented Lagrangian is somewhat lengthy, and so is deferred to the supplement. Projection of the variances onto the constraint set is a simple thresholding operation.

## 5 Experimental Results

### 5.1 Discrete Markov Random Fields

We consider binary Ising models, with variables arranged in NxN lattices with toroidal boundary conditions. Potentials are parametrized as in [19], so that

$$\psi_s = \begin{bmatrix} \exp(h_s) \\ \exp(-h_s) \end{bmatrix}, \quad \psi_{st} = \begin{bmatrix} \exp(J_{st}) & \exp(-J_{st}) \\ \exp(-J_{st}) & \exp(J_{st}) \end{bmatrix}.$$

We sample 500 instances at random from a 10x10 toroidal lattice with each $J_{st} \sim N(0, 1)$ and $h_s \sim N(0, 0.01)$. LBP is run for a maximum of 1000 iterations, and MoM is initialized with a single iteration of LBP. We report average $L_1$ error of the approximate marginals as compared to the true marginals computed with the junction tree algorithm [20]. Marginal errors are reported in Figure 2(a,top), and there is a clear improvement over LBP in the majority of cases.

Direct evaluation of the Bethe free energy does not take into account constraint violations for non-convergent LBP runs. The augmented Lagrangian penalizes constraint violation, but requires a penalty parameter which LBP does not provide. For an objective comparison, we construct a penalized Bethe free energy by evaluating the augmented Lagrangian with fixed penalty $c = 1$ and multipliers $\lambda = 0$. We evaluate this objective at the final iteration of both algorithms. As we see in Figure 2(a,bottom), MoM finds a lower free energy for most trials.

Our implementations of LBP and MoM are in Matlab, and emphasize correctness over efficiency. Nevertheless, computation time for LBP exceeds that of MoM. Wall clock time is measured in seconds across various trials, and the percentiles for LBP are 25%: 1040.46, 50%: 1042.57, and 75%: 1044.85. For MoM they are 25%: 290.25, 50%: 381.62, and 75%: 454.52.

### 5.2 Gaussian Markov Random Fields

For the Gaussian case we again sample 500 random instances from a 10x10 lattice with toroidal boundary conditions. We randomly sample only pairwise normalizable instances and initialization is provided with a single iteration of Gaussian LBP. We find that MoM is generally insensitive to initialization in this model. True marginals are computed by explicitly inverting the model precision matrix and average symmetric $L_1$ error with respect to truth is reported in Figure 2(b,top).

For pairwise normalizable models, Gaussian LBP is guaranteed to converge to the unique fixed point of the Bethe free energy, so it is reassuring that MoM optimization matches LBP performance. The value of the augmented Lagrangian at the final iteration is shown in Figure 2(b,bottom) and again shows that MoM matches Gaussian LBP on pairwise normalizable models. Computation time for MoM is slightly faster with median wall clock time of 58.76 seconds as compared to 103.17 seconds for LBP. The 25% and 75% percentiles are 37.81 and 92.10 seconds for MoM compared to 88.40 and 125.59 seconds for LBP.

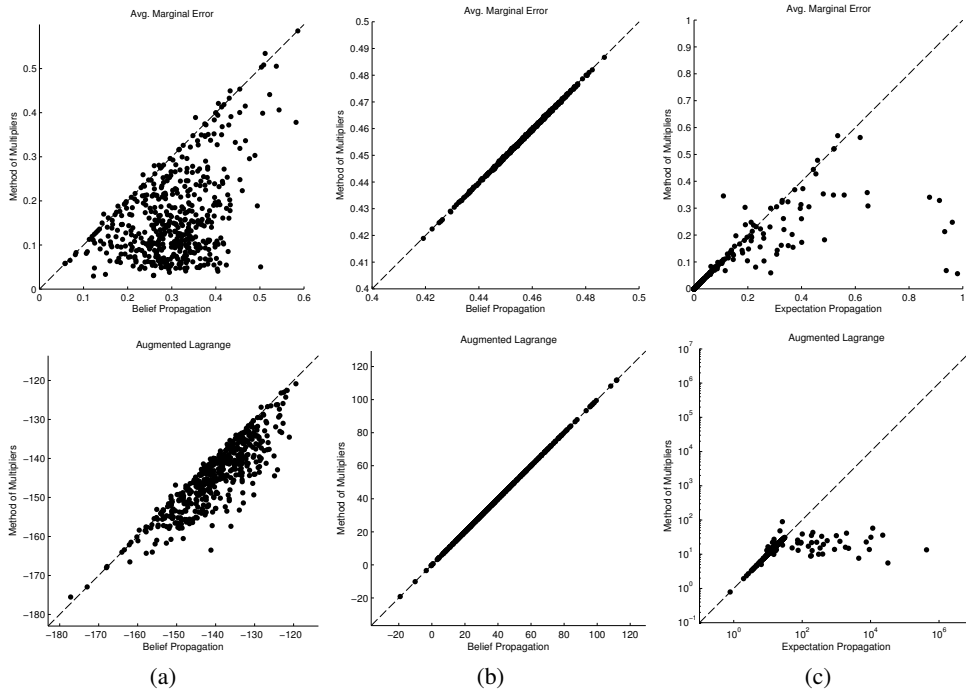

Figure 2: Performance of MoM and LBP on randomly generated (a) discrete $10 \times 10$ toroidal Ising MRFs, (b) $10 \times 10$ toroidal Gaussian MRFs, and (c) clutter models with $N = 30$ observations. Each point corresponds to a single model instance. *Top:* $L_1$ error between estimated and true marginal distributions, averaged over all nodes. *Bottom:* Penalized Bethe free energy constructed by setting $\lambda = 0$, $c = 1$ in the augmented Lagrangian.

## 5.3   Discrete Mixtures of Gaussian Potentials

To test the benefits of avoiding degenerate marginals, we consider the clutter model of Sec. 4.3 with $\mu_0 = 0$, $P_0 = 100$ and $\beta_0 = 0.25$. The variance of the clutter distribution is $\sigma_0^2 = 10$, and of the target distribution $\sigma_1^2 = 1$. We sample $N = 30$ observations for each trial instance.

A good initialization of the multipliers is critical to performance of MoM. We generate 10 initializations by running 5 iterations of EP, each with a different random message update schedule, compute the corresponding Lagrange multipliers for each, and use the one with the lowest value of the augmented Lagrangian. Similarly, we measure EP's performance by the best performing of 10 longer runs. Both methods are run for a maximum of 1000 iterations, and true marginals are computed numerically by finely discretizing the scalar target $x$.

We sample 500 random instances and report average $L_1$ error with respect to true marginals in Figure 2(c,top). We see a significant improvement in the majority of runs. Similarly, the augmented Lagrangian comparison is shown in Figure 2(c,bottom) and MoM often finds a better penalized free energy. While MoM and EP can both suffer from local optima, MoM avoids non-convergence and the output of invalid (negative variance) marginal distributions. Median wall clock time for EP is 0.59 seconds, and 9.80 seconds for MoM. The 25% and 75% percentiles are 0.42 and 0.84 seconds for EP and 0.51 and 49.19 seconds for MoM.

## 6   Discussion

We have proposed an approach for directly minimizing the Bethe variational problem motivated by successful methods in nonlinear programming. Our approach is unique in that we do not relax the constraint on normalizability of the marginals, rather we explicitly enforce it at all points in the optimization. This method directly avoids the creation of degenerate distributions — for example with negative variance — which frequently occur in more greedy approaches for minimizing the Bethe free energy. In addition we obtain convergence guarantees under broadly applicable assumptions.

# References

[1] J.S. Yedidia, W.T. Freeman, and Y. Weiss. Constructing free-energy approximations and generalized Belief Propagation algorithms. *Information Theory, IEEE Transactions on*, 51(7):2282–2312, 2005.

[2] M. J. Wainwright and M. I. Jordan. Graphical models, exponential families, and variational inference. Technical report, UC Berkeley, Dept. of Statistics, 2003.

[3] T. P. Minka. Expectation Propagation for approximate Bayesian inference. *Uncertainty in Artificial Intelligence*, 17:362–369, 2001.

[4] Tom Heskes, Wim Wiegerinck, Ole Winther, and Onno Zoeter. Approximate inference techniques with expectation constraints. *Journal of Statistical Mechanics: Theory and Experiment*, page 11015, 2005.

[5] Dmitry M. Malioutov, Jason K. Johnson, and Alan S. Willsky. Walk-sums and Belief Propagation in Gaussian graphical models. *Journal of Machine Learning Research*, 7:2031–2064, 2006.

[6] B. Cseke and T. Heskes. Properties of bethe free energies and message passing in Gaussian models. *Journal of Artificial Intelligence Research*, 41(2):1–24, 2011.

[7] A. Yuille. CCCP algorithms to minimize the Bethe and Kikuchi free energies: Convergent alternatives to Belief Propagation. *Neural Computation*, 14:1691–1722, 2002.

[8] B. Kappen T. Heskes, K. Albers. Approximate inference and constrained optimization. *Uncertainty in Artificial Intelligence*, 13:313–320, 2003.

[9] Martin J. Wainwright, Tommi S. Jaakkola, and Alan S. Willsky. Tree-reweighted Belief Propagation algorithms and approximate ML estimation by pseudo-moment matching. In *In AISTATS*, 2003.

[10] M. Welling and Y.W. Teh. Belief optimization for binary networks: A stable alternative to Loopy Belief Propagation. In *Uncertainty in Artificial Intelligence*, 2001.

[11] T. Heskes and O. Zoeter. Expectation Propagation for approximate inference in dynamic Bayesian networks. *Uncertainty in Artificial Intelligence*, 18:216–223, 2002.

[12] T. Minka. The EP energy function and minimization schemes. Technical report, MIT Media Lab, 2001.

[13] D.P. Bertsekas. *Nonlinear programming*. Athena Scientific, 1999.

[14] M. Seeger. Bayesian inference and optimal design for the sparse linear model. *Journal of Machine Learning Research*, 9:759–813, 2008.

[15] C. Rasmussen. *Gaussian Processes for Machine Learning*. MIT Press, 2006.

[16] M. Schmidt, E. Van Den Berg, M. Friedlander, and K. Murphy. Optimizing costly functions with simple constraints: A limited-memory projected quasi-Newton algorithm. In *AI & Statistics*, 2009.

[17] D.P. Bertsekas. Constrained optimization and Lagrange multiplier methods. *Computer Science and Applied Mathematics, Boston: Academic Press, 1982*, 1, 1982.

[18] T. Heskes. On the uniqueness of loopy belief propagation fixed points. *Neural Computation*, 16(11):2379–2413, 2004.

[19] J.S. Yedidia, W.T. Freeman, and Y. Weiss. Generalized Belief Propagation. *Advances in neural information processing systems*, pages 689–695, 2001.

[20] Joris M. Mooij. libDAI: A free and open source C++ library for discrete approximate inference in graphical models. *Journal of Machine Learning Research*, 11:2169–2173, August 2010.

